# Recovering Articulated Model Topology from Observed Rigid Motion

**Leonid Taycher, John W. Fisher III, and Trevor Darrell**
Artificial Intelligence Laboratory
Massachusetts Institute of Technology
Cambridge, MA, 02139
{lodrion, fisher, trevor}@ai.mit.edu

## Abstract

Accurate representation of articulated motion is a challenging problem for machine perception. Several successful tracking algorithms have been developed that model human body as an articulated tree. We propose a learning-based method for creating such articulated models from observations of multiple rigid motions. This paper is concerned with recovering topology of the articulated model, when the rigid motion of constituent segments is known. Our approach is based on finding the Maximum Likelihood tree shaped factorization of the joint probability density function (PDF) of rigid segment motions. The topology of graphical model formed from this factorization corresponds to topology of the underlying articulated body. We demonstrate the performance of our algorithm on both synthetic and real motion capture data.

## 1 Introduction

Tracking human motion is an integral part of many proposed human-computer interfaces, surveillance and identification systems, as well as animation and virtual reality systems. A common approach to this task is to model the body as a kinematic tree, and reformulate the problem as articulated body tracking[6]. Most of the state-of-the-art systems rely on predefined kinematic models [16]. Some methods require manual initialization, while other use heuristics [12], or predefined protocols [10] to adapt the model to observations.

We are interested in a principled way to recover articulated models from observations. The recovered models may then be used for further tracking and/or recognition. We would like to approach model estimation as a multistage problem. In the first stage the rigidly moving segments are tracked independently; at the second stage, the topology of the body (the connectivity between the segments) is recovered. After the topology is determined, the joint parameters may be determined.

In this paper we concentrate on the second stage of this task, estimating the underlying topology of the observed articulated body, when the motion of the constituent rigid bodies is known. We approach this as a learning problem, in the spirit of [17]. If we assume that the body may be modeled as a kinematic tree, and motion of a particular rigid segment is known, then the motions of the rigid segments that are connected through that segment are independent of each other. That is, we can model a probability distribution of the full body-

pose as a tree-structured graphical model, where each node corresponds to pose of a rigid segment. This observation allows us to formulate the problem of recovering topology of an articulated body as finding the tree-shaped graphical model that best (in the Maximum Likelihood sense) describes the observations.

## 2   Prior Work

While state-of-the-art tracking algorithms [16] do not address either model creation or model initialization, the necessity of automating these two steps has been long recognized.

The approach in [10] required a subject to follow a set of predefined movements, and recovered the descriptions of body parts and body topology from deformations of apparent contours. Various heuristics were used in [12] to adapt an articulated model of known topology to 3D observations. Analysis of magnetic motion capture data was used by [14] to recover limb lengths and joint locations for known topology, it also suggested similar analysis for topology extraction. A learning based approach for decomposing a set of observed marker positions and velocities into sets corresponding to various body parts was described in [17]. Our work builds on the latter two approaches in estimating the topology of the articulated tree model underlying the observed motion.

Several methods have been used to recover multiple rigid motions from video, such as factorization [3, 18], RANSAC [7], and learning based methods [9]. In this work we assume that the 3-D rigid motions has been recovered and are represented using a 2-D Scaled Prismatic Model (SPM).

## 3   Representing Pose and Motion

A 2-D Scaled Prismatic Model (SPM) was proposed by [15] and is useful for representing image motion of projections of elongated 3-D objects. It is obtained by orthographically "projecting" the major axis of the object to the image plane. The SPM has four degrees of freedom: in-plane translation, rotation, and uniform scale. 3-D rigid motion of an object, may be simulated by SPM transformations, using in-plane translation for rigid translation, and rotation and uniform scaling for plane-parallel and out-of-plane rotations respectively.

SPM motion (or pose) may be expressed as a linear transformation in projective space as

$$\mathbf{M} = \begin{pmatrix} a & -b & e \\ b & a & f \\ 0 & 0 & 1 \end{pmatrix} \tag{1}$$

Following [13] we have chosen to use exponential coordinates, derived from constant velocity equations, to parameterize motion.

An SPM transformation may be represented as an exponential map

$$\mathbf{M} = e^{\hat{\xi}} \qquad \hat{\xi} = \theta \begin{pmatrix} c & -\omega & v_x \\ \omega & c & v_y \\ 0 & 0 & 0 \end{pmatrix} \qquad \xi = \theta \begin{pmatrix} v_x \\ v_y \\ \omega \\ c \end{pmatrix} \tag{2}$$

In this representation $v_x$ is a horizontal velocity, $v_y$ – vertical velocity, $\omega$ – angular velocity, and $c$ is a rate of scale change. $\theta$ is analogous to time parameter. Note that there is an inherent scale ambiguity, since $\theta$ and $(v_x, v_y, \omega, c)^T$ may be chosen arbitrarily, as long as $e^{\hat{\xi}} = \mathbf{M}$.

It can be shown ([13]) that if the SPM transformation is a combination of scaling and rotation, it may be expressed by the sum of two twists, with coincident centers $(u_x, u_y)^T$ of rotation and expansion.

$$\xi = \omega \begin{pmatrix} u_y \\ -u_x \\ 1 \\ 0 \end{pmatrix} + c \begin{pmatrix} -u_x \\ -u_y \\ 0 \\ 1 \end{pmatrix} = \begin{pmatrix} -c & \omega & & \\ -\omega & -c & & \\ & & 1 & \\ & & & 1 \end{pmatrix} \begin{pmatrix} u_x \\ u_y \\ \omega \\ c \end{pmatrix} \tag{3}$$

While "pure" translation, rotation or scale have intuitive representation with twists, the combination or rotation and scale does not. We propose a *scaled twist* representation, that preserves the intuitiveness of representation for all possible SPM motions. We want to separate the "direction" of motion (the direction of translation or the relative amounts of rotation and scale) from the amount of motion.

If the transformation involves rotation and/or scale, then we choose $\theta$ so that $||(\omega, c)||^2 = 1$, and then use eq. 3 to compute the center of rotation/expansion. The computation may be expressed as a linear transformation:

$$\tau = \begin{pmatrix} \theta \\ u_x \\ u_y \\ \omega \\ c \end{pmatrix} = \begin{pmatrix} \sqrt{\tilde{\omega}^2 + \tilde{c}^2} & & & & \\ & -\frac{\tilde{c}}{\tilde{\omega}^2+\tilde{c}^2} & -\frac{\tilde{\omega}}{\tilde{\omega}^2+\tilde{c}^2} & & \\ & \frac{\tilde{\omega}}{\tilde{\omega}^2+\tilde{c}^2} & -\frac{\tilde{c}}{\tilde{\omega}^2+\tilde{c}^2} & & \\ & & & \frac{1}{\sqrt{\tilde{\omega}^2+\tilde{c}^2}} & \\ & & & & \frac{1}{\sqrt{\tilde{\omega}^2+\tilde{c}^2}} \end{pmatrix} \begin{pmatrix} 1 \\ \tilde{v}_x \\ \tilde{v}_y \\ \tilde{\omega} \\ \tilde{c} \end{pmatrix} \tag{4}$$

where $\xi = (\tilde{v}_x, \tilde{v}_y, \tilde{\omega}, \tilde{c})^T$.

The the pure translational motion ($\omega = c = 0$) may be regarded as an infinitely small rotation about a point at infinity, e.g. the translation by $l$ in the direction $(u_x, u_y)$ may be represented as $\tau = \lim_{\omega \to 0}(l|\omega|, \frac{-u_y}{\omega}, \frac{u_x}{\omega}, \omega, 0)^T$, but we choose a direct representation

$$\tau = \begin{pmatrix} \theta \\ u_x \\ u_y \\ 0 \\ 0 \end{pmatrix} = \begin{pmatrix} \sqrt{\tilde{v}_x^2 + \tilde{v}_y^2} & & & & \\ & \frac{1}{\sqrt{\tilde{v}_x^2+\tilde{v}_y^2}} & & & \\ & & \frac{1}{\sqrt{\tilde{v}_x^2+\tilde{v}_y^2}} & & \\ & & & 1 & \\ & & & & 1 \end{pmatrix} \begin{pmatrix} 1 \\ \tilde{v}_x \\ \tilde{v}_y \\ \tilde{\omega} \\ \tilde{c} \end{pmatrix} \tag{5}$$

In both cases $\tau = A(1, \tilde{\xi}^T)^T$, and

$$\det(A) = \begin{cases} \theta^{-3} & \omega \neq 0 \vee c \neq 0 \text{ (rotation/scaling)} \\ \theta^{-1} & \omega = 0 \wedge c = 0 \text{ (pure translation)} \end{cases} \tag{6}$$

Note that $\tau_I = (0, u_x, u_y, \omega, c)^T$ represents identity transformation for any $u_x, u_y, \omega$, and $c$. It is always reported as $\tau_I = \mathbf{0}$.

## 4 Learning Articulated Topology

We wish to infer the underlying topology of an articulated body from noisy observations of a set of rigid body motions. Towards that end we will adopt a statistical framework for fitting a joint probability density. As a practical matter, one must make choices regarding density models; we discuss one such choice although other choices are also suitable.

We denote the set of observed motions of $N$ rigid bodies at time $t, 1 \leq t \leq F$ as a set $\{\mathbf{M}_s^t | 1 \leq s \leq N\}$. Graphical models provide a useful methodology for expressing the dependency structure of a set of random variables (cf. [8]). Variables $M_i$ with observations

$\{\mathbf{M}_i^t | 1 \leq t \leq F\}$ are assigned to the vertices of a graph, while edges between nodes indicate dependency. We shall denote presence or absence of an edge between two variables, $M_i$ and $M_j$ by an index variable $E_{ij}$, equal to one if an edge is present and zero otherwise. Furthermore, if the corresponding graphical model is a spanning tree, it can be expressed as a product of conditional densities (e.g. see [11])

$$P_M(M_1, \ldots, M_N) = \prod_{M_s} P_{M_s | pa(M_s)}(M_s | pa(M_s)) \qquad (7)$$

where pa($M_s$) is the parent of $M_s$. While multiple nodes may have the same parent, each individual node has only one parent node. Furthermore, in any decomposition one node (the root node) has no parent. Any node (variable) in the model can serve as the root node [8]. Consequently, a tree model constrains $E$. Of the possible tree models (choices of $E$), we wish to choose the maximum likelihood tree which is equivalent to the minimum entropy tree [4]. The entropy of a tree model can be written

$$H(M) = \sum_s H(M_s) - \sum_{E_{ij}=1} I(M_i; M_j) \qquad (8)$$

where $H(M_s)$ is the marginal entropy of each variable and $I(M_i; M_j)$ is the mutual information between nodes $M_i$ and $M_j$ and quantifies their statistical dependence. Consequently, the minimum entropy tree corresponds to the choice of $E$ which minimizes the sum of the pairwise mutual informations [1]. The tree denoted by $E$ can be found via the maximum spanning tree algorithm [2] using $I(M_i; M_j)$ for all $i, j$ as the edge weights.

Our conjecture is that if our data are sampled from a variety of motions the topology of the estimated density model is likely to be the same as the topology of the articulated body model. It follows from the intuition that when considering only pairwise relationships, the relative motions of physically connected bodies will be most strongly related.

### 4.1 Estimation of Mutual Information

Computing the minimum entropy spanning tree requires estimating the pairwise mutual informations between rigid motions $\mathbf{M}_i$ and $\mathbf{M}_j$ for all $i, j$ pairs. In order to do so we must make a choice regarding the parameterization of motion and a probability density over that parameterization; to estimate articulated topology it is sufficient to use the the Scaled Prismatic Model with twist parameterization described in Section 3).

### 4.2 Estimating Motion Entropy

We parameterize rigid motion, $\mathbf{M}_i^t$, by the vector of quantities $\xi_i^t$ (cf. Eq. 2). In general,

$$H(\mathbf{M}_i) \neq H(\xi_i), \qquad (9)$$

but since there is a one-to-one correspondence between the $\mathbf{M}_i$'s and $\xi_i$'s [4], we can estimate the $I(\mathbf{M}_i; \mathbf{M}_j)$ by first computing $\xi_i^t, \xi_j^t$ from $\mathbf{M}_i^t, \mathbf{M}_j^t$

$$I(\mathbf{M}_i; \mathbf{M}_j) = I(\xi_i; \xi_j) = H(\xi_j) - H(\xi_j | \xi_i) \qquad (10)$$

Furthermore, if the relative motion $M_{j|i}$ between segments $s_i$ and $s_j$ ($M_j^t = M_i^t M_{j|i}^t$) is assumed to be independent of $M_i$, it can be shown that

$$H(\xi_j | \xi_i) = H(\log M_i M_{j|i} | \log M_i) = H(\log M_{j|i}) = H(\xi_{j|i}). \qquad (11)$$

We wish to use scaled twists (Section 3) to compute the entropies involved. Since the involved quantities are in the linear relationship $\tau = A(1, \tilde{\xi}^T)^T$ (Eqs. 4 and 5), the entropies are related,

$$H(\xi) = H(\tau) - E[\log \det(A)], \qquad (12)$$

where $E[\log \det(A)]$ may be estimated using Equation 6.

### 4.3 Estimating the Motion Kernel

In order to estimate the entropy of motion, we need to estimate the probability density based on the available samples. Since the functional form of the underlying density is not known we have chosen to use kernel-based density estimator,

$$\hat{p}(\tau) = \alpha \sum_i K(\tau; \tau_i). \tag{13}$$

Since our task is to determine the articulated topology, we wish to concentrate on "spatial" features of the transformation, center of rotation for rotational motion, and the direction of translation for translational, that correspond to two common kinds of joints, spherical and prismatic. Thus we need to define a kernel function $K(\tau_1; \tau_2)$ that captures the following notion of "distance" between the motions:

1. If $\tau_1$ and $\tau_2$ do not represent pure translational motions, then they should be considered to be close if their centers of rotation are close.

2. If $\tau_1$ and $\tau_2$ are pure translations, then they should be considered close if their directions are close.

3. If $\tau_1$ and $\tau_2$ represent different types of motion (i.e. rotation/scale vs. translation), then they are arbitrarily far apart.

4. The identity transformation ($\theta = 0$) is equidistant from all possible transformations (since any $(u_x, u_y, \omega, c)^T$ combined with $\theta = 0$ produces identity)

One kernel that satisfies these requirements is the following:

$$K(\tau_1; \tau_2) = \begin{cases} K_R((u_{x1}, u_{y1}); (u_{x2}, u_{y2})) & \text{Condition 1} \\ & (\omega_1 \neq 0 \lor c_1 \neq 0) \land (\omega_2 \neq 0 \lor c_2 \neq 0) \\ K_T((u_{x1}, u_{y1}); (u_{x2}, u_{y2})) & \text{Condition 2} \\ & \omega_1 = 0 \land c_1 = 0 \land \omega_2 = 0 \land c_2 = 0 \\ 0 & \text{Condition 3} \\ & (\omega_1 \neq 0 \lor c_1 \neq 0) \land (\omega_2 = 0 \land c_2 = 0) \\ 0 & \text{Condition 3} \\ & (\omega_1 = 0 \land c_1 = 0) \land (\omega_2 \neq 0 \lor c_2 \neq 0) \\ \delta(0) & \text{Condition 4.} \\ & \theta_1 = 0 \lor \theta_2 = 0 \end{cases} \tag{14}$$

where $K_R$ and $K_T$ are Gaussian kernels with covariances estimated using methods from [5].

## 5 Implementation

The input to our algorithm is a set of SPM poses (Section 3) $\{\mathbf{P}_s^t | 1 \leq s \leq S, 1 \leq t \leq T\}$, where $S$ is the number of tracked rigid segments and $F$ is the number of frames. In order to compute the mutual information between the motion of segments $s_1$ and $s_2$, we first compute motions of segment $s_1$ in frames $1 < t \leq F$ relative to its position in frame $t_1 = 1$,

$$\mathbf{M}_{s_1}^{t_1 t} = \mathbf{P}_{s_1}^t (\mathbf{P}_{s_1}^{t_1})^{-1}, \tag{15}$$

and the transformation of $s_2$ relative to $s_1$ (with the relative pose $\mathbf{P}_{s_2|s_1} = (\mathbf{P}_{s_1})^{-1} \mathbf{P}_{s_2}$),

$$\mathbf{M}_{s_2|s_1}^{t_1 t} = ((\mathbf{P}_{s_1}^t)^{-1} \mathbf{P}_{s_2}^t)((\mathbf{P}_{s_1}^{t_1})^{-1} . \mathbf{P}_{s_2}^{t_1})^{-1} \tag{16}$$

The parameter vectors $\tau_{s_2}^{t_1 t}$ and $\tau_{s_2|s_1}^{t_1 t}$ are then extracted from the transformation matrices $\mathbf{M}_{s_2}$ and $\mathbf{M}_{s_2|s_1}$ (cf. Section 3), and the mutual information is estimated as described in Section 4.2.

# 6 Results

We have tested our algorithm both on synthetic and motion capture data. Two synthetic sequences were generated with the following steps. First, the rigid segments were positioned by randomly perturbing parameters of the corresponding kinematic tree structure. A set of feature points was then selected for each segment. At each time step point positions were computed based on the corresponding segment pose, and perturbed with Gaussian noise with zero mean and standard deviation of 1 pixel. The inputs to the algorithm were the segment poses re-estimated from the feature point coordinates. In the motion capture-based experiment, the segment poses were estimated from the marker positions.

The results of the experiments are shown in the Figures 6.1, 6.2 and 6.3. The first experiment involved a simple kinematic chain with 3 segments in order to demonstrate the operation of the algorithm. The system has a rotational joint between $S_1$ and $S_2$ and prismatic joint between $S_2$ and $S_3$.

The sample configurations of the articulated body are shown in the first row of the Figures 6.1. The graph computed using method from Section 4.2 and the corresponding maximum spanning tree are in Figures 6.1(d, e).

The second experiment involved a humanoid torso-like synthetic model containing 5 rigid segments. It was processed in a way similar to the first experiment. The results are shown in Figure 6.2.

For the human motion experiment, we have used motion capture data of a dance sequence (Figure 6.3(a-c)). The rigid segment motion was extracted from the positions of the markers tracked across 220 frames (the marker correspondence to the body locations was known). The algorithm was able to correctly recover the articulated body topology (Compare Figures 6.3(e) and 6.3(a)), when provided only with the extracted segment poses. The dance is a highly structured activity, so not all degrees of freedom were explored in this sequence, and mutual information between some unconnected segments (e.g. thighs $S_3$ and $S_7$) was determined to be relatively large, although this did not impact the final result.

# 7 Conclusions

We have presented a novel general technique for recovering the underlying articulated structure from information about rigid segment motion. Our method relies on only a very weak assumption, that this structure may be represented by a tree with unknown topology. While the results presented in this paper were obtained using the Scaled Prismatic Model and non-parametric density estimator, our methodology does not rely on either modeling assumption.

## References

[1] C. K. Chow and C. N. Liu. Approximating discrete probability distributions with dependence trees. *IEEE Transactions on Information Theory*, IT-14(3):462–467, May 1968.

[2] Thomas H. Cormen, Charles E. Leiserson, and Ronald L. Rivern. *Introduction to Algorithms*. MIT Press, Cambridge, MA, 1990.

[3] Joao Paolo Costeira and Takeo Kanade. A multibody factorization method for independently moving objects. *International Journal of Computer Vision*, 29(3):159–179, 1998.

[4] T. M. Cover and J. A. Thomas. *Elements of Information Theory*. John Wiley & Sons, Inc., New York, 1991.

[5] Luc Devroye. *A Course in Density Estimation*, volume 14 of *Progress in Probability and Statistics*. Birkhauser, Boston, 1987.

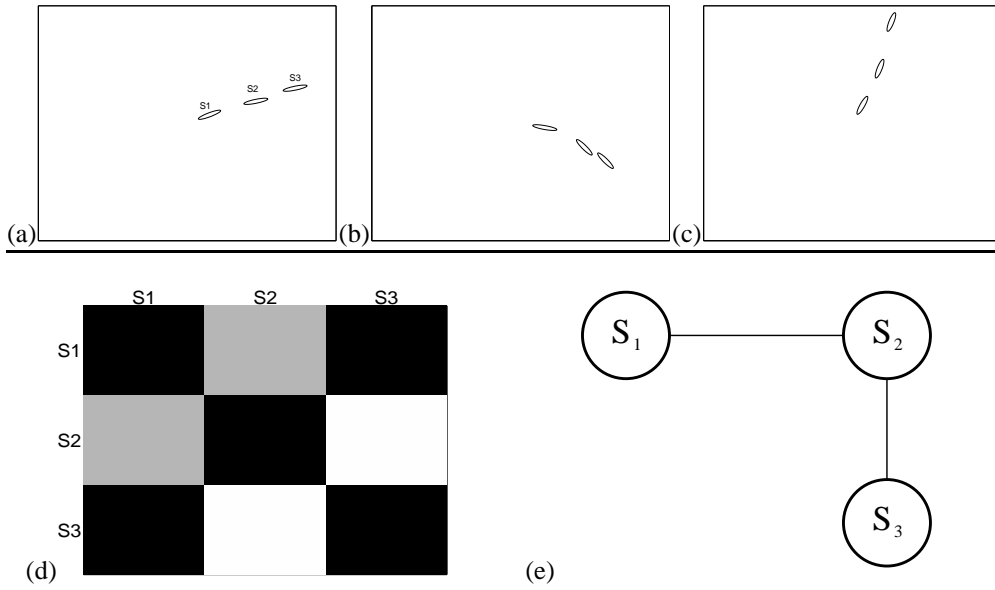

Figure 6.1: Simple kinematic chain topology recovery. The first row shows 3 sample frames from a 100 frame synthetic sequence. The adjacency matrix of the mutual information graph is shown in (d), with intensities corresponding to edge weights. The vertices in the graph correspond to the rigid segments labeled in (a). (e) is the recovered articulated topology.

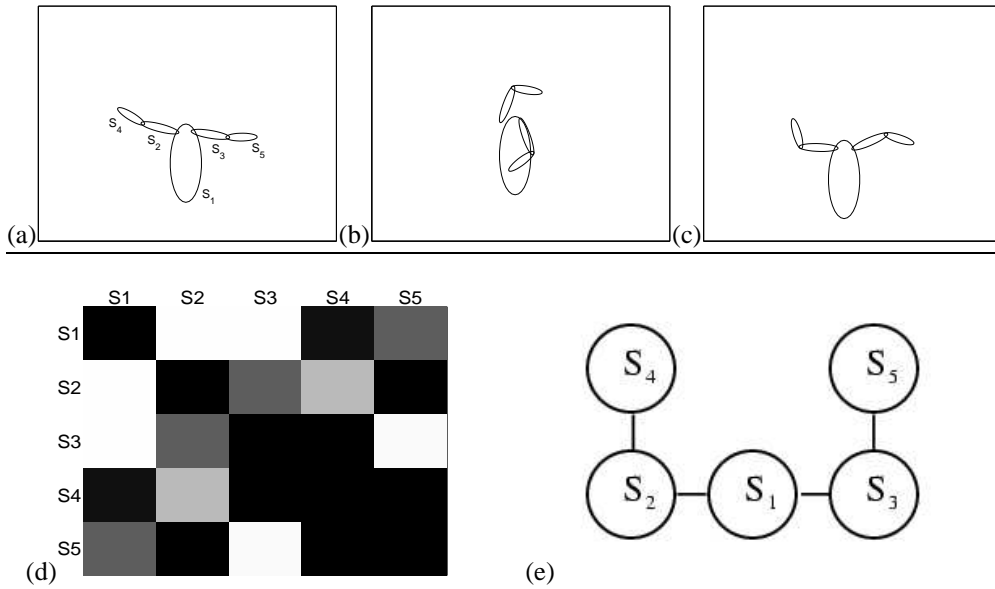

Figure 6.2: Humanoid torso synthetic test. The sample frames from a randomly generated 150 frame sequence are shown in (a), (b), and (c). The adjacency matrix of the mutual information graph is shown in (d), with intensities corresponding to edge weights. The vertices in the graph correspond to the rigid segments labeled in (a). (e) is the recovered articulated topology.

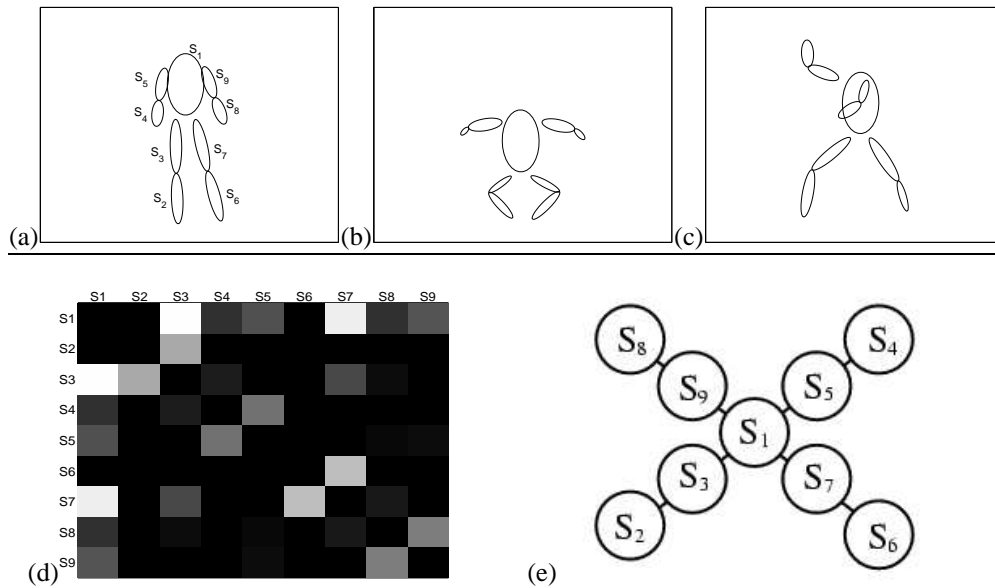

Figure 6.3: Motion Capture based test. (a), (b), and (c) are the sample frames from a 220 frame sequence. The adjacency matrix of the mutual information graph is shown in (d), with intensities corresponding to edge weights. The vertices in the graph correspond to the rigid segments labeled in (a). (e) is the recovered articulated topology.

[6] David C. Hogg. Model-based vision: A program to see a walking person. *Image and Vision Computing*, 1(1):5–20, 1983.

[7] Yi-Ping Hung, Cheng-Yuan Tang, Sheng-Wen Shin, Zen Chen, and Wei-Song Lin. A 3d feature-based tracker for tracking multiple moving objects with a controlled binocular head. Technical report, Academia Sinica Institute of Information Science, 1995.

[8] Finn Jensen. *An Introduction to Bayesian Networks*. Springer, 1996.

[9] N. Jojic and B.J. Frey. Learning flexible sprites in video layers. In *Computer Vision and Pattern Recognition*, pages I:199–206, 2001.

[10] Ioannis A. Kakadiaris and Dimirti Metaxas. 3d human body acquisition from multiple views. In *Proc. Fifth International Conference on Computer Vision*, pages 618–623, 1995.

[11] Marina Meila. *Learning Mixtures of Trees*. PhD thesis, MIT, 1998.

[12] Ivana Mikic, Mohan Triverdi, Edward Hunter, and Pamela Cosman. Articulated body posture estimation from multi-camera voxel data. In *Computer Vision and Pattern Recognition*, 2001.

[13] Richard M. Murray, Zexiang Li, and S. Shankar Sastry. *A Mathematical Introduction to Robotic Manipulation*. CRC Press, 1994.

[14] J. O'Brien, R. E. Bodenheimer, G. Brostow, and J. K. Hodgins. Automatic joint parameter estimation from magnetic motion capture data. In *Graphics Interface'2000*, pages 53–60, 2000.

[15] James M. Regh and Daniel D. Morris. Singularities in articulated object tracking with 2-d and 3-d models. Technical report, Digital Equipment Corporation, 1997.

[16] Hedvig Sidenbladh, Michael J. Black, and David J. Fleet. Stochastic tracking of 3d human figures using 2d image motion. In *Proc. European Conference on Computer Vision*, 2000.

[17] Yang Song, Luis Goncalves, Enrico Di Bernardo, and Pietro Perona. Monocular perception of biological motion - detection and labeling. In *Proc. International Conference on Computer Vision*, pages 805–812, 1999.

[18] Ying Wu, Zhengyou Zhang, Thomas S. Huang, and John Y. Lin. Multibody grouping via orthogonal subspace decomposition. In *Proc. IEEE Conf. on Computer Vision and Pattern Recognition*, 2001.
